# Decomposition of Reinforcement Learning for Admission Control of Self-Similar Call Arrival Processes

**Jakob Carlström**

Department of Electrical Engineering, Technion, Haifa 32000, Israel
jakob@ee.technion.ac.il

## Abstract

This paper presents predictive gain scheduling, a technique for simplifying reinforcement learning problems by decomposition. Link admission control of self-similar call traffic is used to demonstrate the technique. The control problem is decomposed into on-line prediction of near-future call arrival rates, and precomputation of policies for Poisson call arrival processes. At decision time, the predictions are used to select among the policies. Simulations show that this technique results in significantly faster learning without any performance loss, compared to a reinforcement learning controller that does not decompose the problem.

## 1 Introduction

In multi-service communications networks, such as Asynchronous Transfer Mode (ATM) networks, resource control is of crucial importance for the network operator as well as for the users. The objective is to maintain the service quality while maximizing the operator's revenue. At the call level, service quality (Grade of Service) is measured in terms of call blocking probabilities, and the key resource to be controlled is bandwidth. Network routing and call admission control (CAC) are two such resource control problems.

Markov decision processes offer a framework for optimal CAC and routing [1]. By modelling the dynamics of the network with traffic and computing control policies using dynamic programming [2], resource control is optimized. A standard assumption in such models is that calls arrive according to Poisson processes. This makes the models of the dynamics relatively simple. Although the Poisson assumption is valid for most user-initiated requests in communications networks, a number of studies [3, 4, 5] indicate that many types of arrival processes in wide-area networks as well as in local area networks are statistically *self-similar*. This makes it difficult to find models of the dynamics, and the models become large and complex. If the number of system states is large, straightforward application of dynamic programming is unfeasible. Nevertheless, the "fractal" burst structure of self-similar traffic should be possible to exploit in the design of efficient resource control methods.

We have previously presented a method based on temporal-difference (TD) learning for CAC of self-similar call traffic, which yields higher revenue than a TD-based controller assuming Poisson call arrival processes [7]. However, a drawback of this method is the slow convergence of the control policy. This paper presents an alternative solution to the above

problem, called *predictive gain scheduling*. It decomposes the control problem into two parts: time-series prediction of near-future call arrival rates and precomputation of a set of control policies for Poisson call arrival processes. At decision time, a policy is selected based on these predictions. Thus, the self-similar arrival process is approximated by a quasi-stationary Poisson process. The rate predictions are made by (artificial) neural networks (NNs), trained on-line. The policies can be computed using dynamic programming or other reinforcement learning techniques [6].

This paper concentrates on the link admission control problem. However, the controllers we describe can be used as building block in optimal routing, as shown in [8] and [9]. Other recent work on reinforcement learning for CAC and routing includes [10], where Marbach et al. show how to extend the use of TD learning to network routing, and [11] where Tong et al. apply reinforcement learning to routing subject to Quality of Service constraints.

## 2    Self-Similar Call Arrival Processes

The limitations of the traditional Poisson model for network arrival processes have been demonstrated in a number of studies, e.g. [3, 4, 5], which indicate the existence of heavy-tailed inter-arrival time distributions and long-term correlations in the arrival processes. Self-similar (fractal-like) models have been shown to correspond better with this traffic.

A self-similar arrival process has no "natural" burst length. On the contrary, its arrival intensity varies considerably over many time scales. This makes the variance of its sample mean decay slowly with the sample size, and its auto-correlation function decay slowly with time, compared to Poisson traffic [4].

The complexity of control and prediction of Poisson traffic is reduced by the memory-less property of the Poisson process: its expected future depends on the arrival intensity, but not on the process history. On the other hand, the long-range dependence of self-similar traffic makes it possible to improve predictions of the process future by observing the history.

A compact statistical measure of the degree of self-similarity of a stochastic process is the *Hurst parameter* [4]. For self-similar traffic this parameter takes values in the interval (0.5, 1], whereas Poisson processes have a Hurst parameter of 0.5.

## 3    The Link Admission Control Problem

In the link admission control (LAC) problem, a link with capacity $C$ [units/s] is offered calls from $K$ different service classes. Calls belonging to such a class $j \in J = \{1, ..., K\}$ have the same bandwidth requirements $b_j$ [units/s]. The per-class call holding times are assumed to be exponentially distributed with mean $1/\mu_j$ [s].

Access to the link is controlled by a *policy* $\pi$ that maps *states* $x \in X$ to *actions* $a \in A$, $\pi: X \rightarrow A$. The set $X$ contains all feasible link states, and the action set is

$$A = \left\{ (a_1, ..., a_K) : a_j \in \{0, 1\}, j \in J \right\},$$

where $a_j$ is 0 for rejecting a presumptive class-$j$ call and 1 for accepting it. The set of link states is given by $X = N \times H$, where $N$ is the set of feasible call number tuples, and $H$ is the Cartesian product of some representations, $h_j$, of the history of the per-class call arrival processes (needed because of the memory of self-similar arrival processes). $N$ is given by

$$N = \left\{ n : n_j \geq 0, j \in J; \sum_{j \in J} n_j b_j \leq C \right\},$$

where $n_j$ is the number of type-$j$ calls accepted on the link.

We assume uniform call charging, which means that the reward rate $\rho(t)$ at time $t$ is equal to the carried bandwidth:

$$\rho(t) = \rho(x(t)) = \sum_{j \in J} n_j(t) b_j \tag{1}$$

Time evolves continuously, with discrete call arrival and departure events, enumerated by $k = 0, 1, 2, \ldots$ Denote by $r_{k+1}$ the immediate reward obtained from entering a state $x_k$ at time $t_k$ until entering the next state $x_{k+1}$ at time $t_{k+1}$. The expectation of this reward is

$$E_\pi\{r_{k+1}\} = E_\pi\{\rho(x_k)[t_{k+1} - t_k]\} = \rho(x_k)\tau(x_k, \pi(x_k)) \tag{2}$$

where $\tau(x_k, \pi)$ is the expected sojourn time in state $x_k$ under policy $\pi$.

By taking optimal actions, the policy controls the probabilities of state transitions so as to increase the probability of reaching states that yield high long-term rewards. The objective of link admission control is to find a policy $\pi$ that maximizes the *average reward per stage:*

$$\bar{R}(\pi) = \lim_{N \to \infty} E_\pi\left\{ \frac{1}{N} \sum_{k=0}^{N} r_{k+1} \;\middle|\; x_0 = x \right\}, \; x \in X. \tag{3}$$

Note that the average reward does not depend on the initial state $x$, as the contribution from this state to the average reward tends to zero as $N \to \infty$ (assuming, for example, that the probability of reaching any other state $y \in X$ from every state $x \in X$ is positive).

Certain states are of special interest for the optimal policy. These are the states that are candidates for *intelligent blocking*. The set of such states $X_{ib} \subset X$ is given by $X_{ib} = N_{ib} \times H$, where $N_{ib}$ is the set of call number tuples for which the available bandwidth is a multiple of the bandwidth of a wideband call. In the states of $X_{ib}$, the long-term reward may be increased by rejecting narrowband calls to reserve bandwidth for future, expected wideband calls.

## 4  Solution by Predictive Gain Scheduling

Gain scheduling is a control theory technique, where the parameters of a controller are changed as a function of operating conditions [12]. The approach taken here is to look up policies in a table from predictions of the near-future per-class call arrival rates.

For Poisson call arrival processes, the optimal policy for the link admission control problem does not depend on the history, $H$, of the arrival processes. Due to the memory-less property, only the (constant) per-class arrival rates $\lambda_j$, $j \in J$, matter. In our gain scheduled control of self-similar call arrival processes, near-future $\lambda_j$ are predicted from $h_j$. The self-similar call arrival processes are approximated by quasi-stationary Poisson processes, by selecting precomputed polices (for Poisson arrival processes) based on predicted $\lambda_j$'s. One radial-basis function (RBF) NN per class is trained to predict its near-future arrival rate.

### 4.1  Solving the Link Admission Control problem for Poisson Traffic

For Poisson call arrival processes, dynamic programming offers well-established techniques for solving the LAC problem [1]. In this paper, policy iteration is used. It involves two steps: *value determination* and *policy improvement*.

The value determination step makes use of the objective function (3), and the concept of *relative values* [1]. The difference $v(x, \pi) - v(y, \pi)$ between two relative values under a policy $\pi$ is the expected difference in accumulated reward over an infinite time interval, starting in state $x$ instead of state $y$. In this paper, the relative values are computed by solving a system of linear equations, a method chosen for its fast convergence. The dynamics of

the system are characterized by state transition probabilities, given by the policy, the per-class call arrival intensities, $\{\lambda_j\}$, and mean holding times, $\{1/\mu_j\}$.

The policy improvement step consists of finding the action that maximizes the relative value at each state. After improving the policy, the value determination and policy improvement steps are iterated until the policy does not change [9].

## 4.2 Determining The Prediction Horizon

Over what future time horizon should we predict the rates used to select policies? In this work, the prediction horizon is set to an average of estimated mean first passage times from states back to themselves, in the following referred to as the *mean return time*. The arrival process is approximated by a quasi-stationary Poisson process within this time interval.

The motivation for this choice of prediction horizon is that the effects of a decision (action) in a state $x_d$ influence the future probabilities of reaching other states and receiving the associated rewards, until the state $x_d$ is reached the next time. When this happens, a new decision can be made, where the previous decision does no longer influence the future expected reward. In accordance with the assumption of quasi-stationarity, the mean return time can be estimated for call tuples $n$ instead of the full state descriptor, $x$.

In case of Poisson call arrival processes, the mean first passage times $E_\pi\{T_{in}\}$ from other states to a state $n$ are the unique solution of the linear system of equations

$$E_\pi\{T_{mn}\} = \tau(m, a) + \sum_{l \in N \setminus \{n\}} E_\pi\{T_{ln}\}, \quad m \in N \setminus \{n\}, \ a = \pi(m) \tag{4}$$

The limiting probability $q_n$ of occupying state $n$ is determined for all states that are candidates for intelligent blocking, by solving a linear system of equations $qB = 0$. $B$ is a matrix containing the state transition intensities, given by $\{\lambda_j\}$ and $\{1/\mu_j\}$.

The mean return time for the link, $T_l$, is defined as the average of the individual mean return times of the states of $N_{ib}$, weighted by their limiting probabilities and normalized:

$$T_l = \sum_{n \in N_{ib}} q_n T_{nn} \bigg/ \sum_{n \in N_{ib}} q_n \tag{5}$$

For ease of implementation, this time window is expressed as a number of call arrivals. The window length $L_j$ for class $j$ is computed by multiplying the mean return time by the arrival rate, $L_j = \lambda_j T_l$, and rounding off to an integer. Although the window size varies with $\lambda_j$, this variation is partly compensated by $T_l$ decreasing with increasing $\lambda_j$.

## 4.3 Prediction of Future Call Arrival Rates

The prediction of future arrival call rates is naturally based on measures of recent arrival rates. In this work, the following representation of the history of the arrival process is used: for all classes $j \in J$, exponentially weighted running averages $h_j = (h_{j1}, ..., h_{jM})$ of the inter-arrival times are computed on different time scales. These *history vectors* are computed using forgetting factors $\{a_1, ..., a_M\}$ taking values in the interval $(0, 1)$:

$$h_{ji}(k) = a_i[t_j(k) - t_j(k - 1)] + (1 - a_i)h_{ji}(k - 1), \tag{6}$$

where $t_j(k)$ is the arrival time of the $k$-th call from class $j$.

In studies of time-series prediction, non-linear feed-forward NNs outperform linear predictors on time series with long memory [13]. We employ RBF NNs with symmetric Gaussian basis functions. The activations of the RBF units are normalized by division by the sum of activations, to produce a smooth output function. The locations and widths of the RBF units can be determined by inspection of the data sets, to cover the region of history vectors.

The NN is trained with the average inter-arrival time as target. After every new call arrival, the prediction error $\epsilon_j(k)$ is computed:

$$\epsilon_j(k) = \frac{1}{L_j} \sum_{i=1}^{L_j} [t(k+i) - t(k+i-1)] - y_j(k). \tag{7}$$

Learning is performed on-line using the least mean squares rule, which means that the updating must be delayed by $L_j$ call arrivals. The predicted per-class arrival rates $\hat{\lambda}_j(k) = y(k)^{-1}$ are used to select a control policy on the arrival of a call request.

Given the prediction horizon and the arrival rate predictor, $\alpha_1, ..., \alpha_M$ can be tuned by linear search to minimize the prediction error on sample traffic traces.

# 5  Numerical study

The performance of the gain scheduled admission controller was evaluated on a simulated link with capacity $C = 24$ [units/s], that was offered calls from self-similar call arrival processes. For comparison, the simulations were repeated with three other link admission controllers: two TD-based controllers, one table-based and one NN based, and a controller using complete sharing, i.e. to accept a call if the free capacity on the link is sufficient.

The NN based TD controller [7] uses RBF NNs (one per $n \in N$), receiving $(h_1, h_2)$ as input. Each NN has 65 hidden units, factorized to 8 units per call class, plus a default activation unit. Its weights were initialized to favor acceptance of all feasible calls in all states.

The table-based TD controller assumes Poisson call arrival processes. From this, it follows that the call number tuples $n \in N$ constitute Markovian states. Consequently, the value function table stores only one value per $n$. This controller was used for evaluation of the performance loss from incorrectly modelling self-similar call traffic by Poisson traffic.

## 5.1  Synthesis of Call Traffic

Synthetic traffic traces were generated from a Gaussian *fractional auto-regressive integrated moving average* model, FARIMA $(0, d, 0)$. This results in a statistically self-similar arrival process, where the Hurst parameter is easily tuned [7].

We generated traces containing arrival/departure pairs from two call classes, characterized by bandwidth requirements $b_1 = 1$ (narrow-band) and $b_2 = 6$ (wide-band) [units/s] and call holding times with mean $1/\mu_1 = 1/\mu_2 = 1$ [s]. A Hurst parameter of 0.85 was used, and the call arrival rates were scaled to make the expected long-term arrival rates $\lambda_1$ and $\lambda_2$ for the two classes fulfill $b_1\lambda_1/\mu_1 + b_2\lambda_2/\mu_2 = 1.25\,C$. The ratio $\lambda_1/\lambda_2$ was varied from 0.4 to 2.0.

## 5.2  Gain Scheduling

For simplicity, a constant prediction horizon was used throughout the simulations. This was computed according to section 4.2. By averaging the resulting prediction windows for $\lambda_1/\lambda_2 = 0.4$, 1.0 and 2.0, a window size $L_1 = L_2 = 6$ was obtained.

The table of policies to be used for gain scheduling was computed for predicted $\hat{\lambda}_1$ and $\hat{\lambda}_2$ ranging from 0.5 to 15 with step size 0.5; in total 900 policies. The two rate-prediction NNs both had 9 hidden units. The NNs' weights were initialized to 0.

## 5.3  Numerical results

Both the TD learning controllers and the gain scheduling controller were allowed to adapt to the first 400 000 simulated call arrivals of the traffic traces. The throughput obtained by all four methods was measured on the subsequent 400 000 call arrivals.

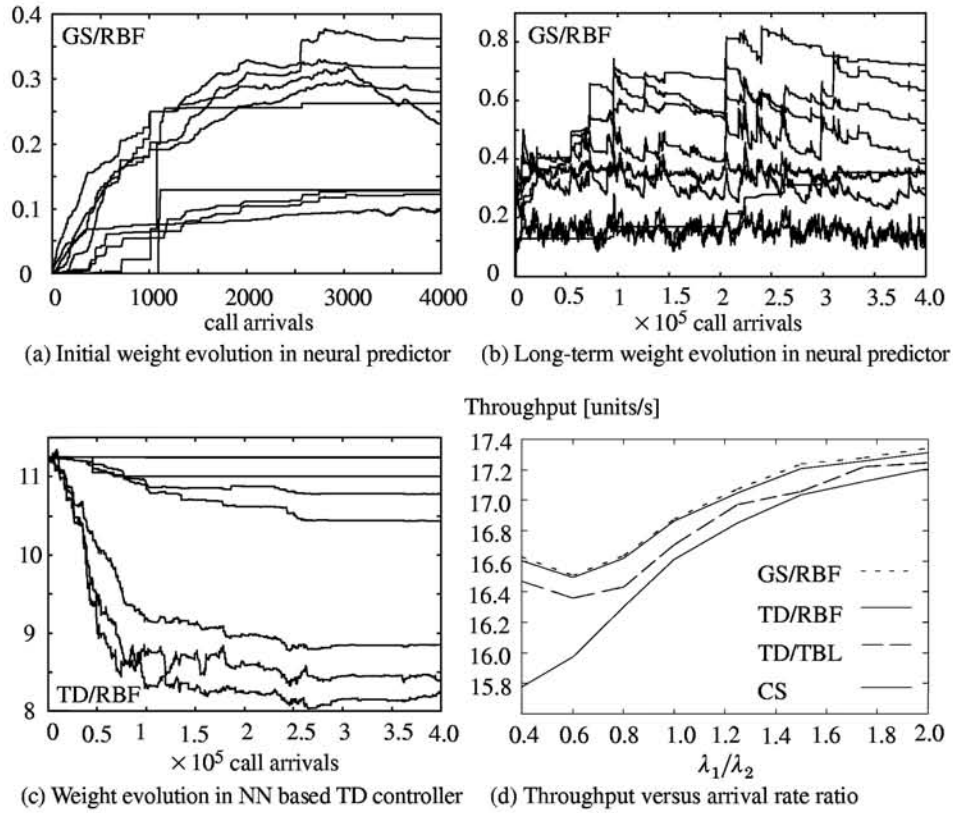

(a) Initial weight evolution in neural predictor    (b) Long-term weight evolution in neural predictor

(c) Weight evolution in NN based TD controller    (d) Throughput versus arrival rate ratio

Figure 1: Weight evolution for NN predictor (a, b); NN based TD-controller (c). Performance (d).

Figure 1 (a, b) shows the evolution of the weights of the call arrival rate predictor for class 2, and figure 1 (c) displays nine weights of the RBF NN corresponding to the call number tuple $(n_1, n_2) = (6, 2)$, which is a candidate for intelligent blocking. These weights correspond to eight different class-2 center vectors, plus the default activation.

The majority of the weights of the gain scheduling RBF NN seems to converge in a few thousand call arrivals, whereas the TD learning controller needs about 100 000 call arrivals to converge. This is not surprising, since the RBF NNs of the TD learning controllers split up the set of training data, so that a single NN is updated much less frequently than a rate-predicting NN in the gain scheduling controller. Secondly, the TD learning NNs are trained on moving targets, due to the temporal-difference learning rule, stochastic action selection and a changing policy.

A few of the weights of the gain scheduling NN change considerably even after long training. These weights correspond to RBF units that are activated by rare, large inputs.

Figure 1 (d) evaluates performance in terms of throughput versus arrival rate ratio. Each data point is the averaged throughput for 10 traffic traces. Gain scheduling (GS/RBF) achieves the same throughput as TD learning with RBF NNs (TD/RBF), up to 1.3% compared to tabular TD learning (TD/TBL), and up to 5.7% better than complete sharing (CS). The difference in throughput between TD learning and complete sharing is greatest for low arrival rate ratios, since the throughput increase by reserving bandwidth for high-rate wideband calls is considerably higher than the loss of throughput from the blocked low-rate narrowband traffic.

# 6 Conclusion

We have presented predictive gain scheduling, a technique for decomposing reinforcement learning problems. Link admission control, a sub-problem of network routing, was used to demonstrate the technique. By predicting near-future call arrival rates from one part of the full state descriptor, precomputed policies for Poisson call arrival processes (computed from the rest of the state descriptor) were selected. This increased the on-line convergence rate approximately 50 times, compared to a TD-based admission controller getting the full state descriptor as input. The decomposition did not result in any performance loss.

The computational complexity of the controller using predictive gain scheduling may reach a computational bottleneck if the size of the state space is increased: the determination of optimal policies for Poisson traffic by policy iteration. This can be overcome by state aggregation [2], or by parametrization the relative value function combined with temporal-difference learning [10]. It is also possible to significantly reduce the number of relative value functions. In [14], we showed that linear interpolation of relative value functions distributed by an error-driven algorithm enables the use of less than 30 relative value functions without performance loss. Further, we have successfully employed gain scheduled link admission control as a building block of network routing [9], where the performance improvement compared to conventional methods is larger than for the link admission control problem.

The use of gain scheduling to reduce the complexity of reinforcement learning problems is not limited to link admission control. In general, the technique should be applicable to problems where parts of the state descriptor can be used, directly or after preprocessing, to select among policies for instances of a simplified version of the original problem.

## References

[1] Z. Dziong, *ATM Network Resource Management*, McGraw-Hill, 1997.

[2] D.P. Bertsekas, *Dynamic Programming and Optimal Control*, Athena Scientific, Belmont, Mass., 1995.

[3] V. Paxson and S. Floyd, "Wide-Area Traffic: The Failure of Poisson Modeling", *IEEE/ACM Transactions on Networking*, vol. 3, pp. 226-244, 1995.

[4] W.E. Leland, M.S. Taqqu, W. Willinger and D.V. Wilson, "On the Self-Similar Nature of Ethernet Traffic (Extended Version)", *IEEE/ACM Transactions on Networking*, vol. 2, no. 1, pp. 1–15, Feb. 1994.

[5] A. Feldman, A.C. Gilbert, W. Willinger and T.G. Kurtz, "The Changing Nature of Network Traffic: Scaling Phenomena", *Computer Communication Review*, vol. 28, no. 2, pp. 5–29, April 1998.

[6] R.S. Sutton and A.G. Barto, *Reinforcement Learning: An Introduction*, MIT Press, Cambridge, Mass., 1998.

[7] J. Carlström and E. Nordström, "Reinforcement Learning for Control of Self-Similar Call Traffic in Broadband Networks", *Teletraffic Engineering in a Competitive World – Proceedings of The 16th International Teletraffic Congress* (ITC 16), pp. 571–580, Elsevier Science B.V., 1999.

[8] Z. Dziong and L. Mason, "Call Admission Control and Routing in Multi-service Loss Networks", *IEEE Transactions on Communications*, vol. 42, no. 2. pp. 2011–2022, Feb. 1994.

[9] J. Carlström and E. Nordström, "Gain Scheduled Routing in Multi-Service Networks", Technical Report 2000-009, Dept. of Information Technology, Uppsala University, Uppsala, Sweden, April 2000.

[10] P. Marbach, O. Mihatsch and J.N. Tsitsiklis, "Call Admission Control and Routing in Integrated Service Networks Using Neuro-Dynamic Programming", *IEEE J. Sel. Areas of Comm.*, Feb. 2000.

[11] H. Tong and T. Brown, "Adaptive Call Admission Control Under Quality of Service Constraints: A Reinforcement Learning Solution", *IEEE Journal on Selected Areas in Communications*, Feb. 2000.

[12] K.J. Åström and B. Wittenmark, *Adaptive Control*, 2nd ed., Addison-Wesley, 1995.

[13] S. Haykin, *Neural Networks: A Comprehensive Foundation*, 2nd ed., Macmillan College Publishing Co., Englewood Cliffs, NJ, 1999.

[14] J. Carlström, "Efficient Approximation of Values in Gain Scheduled Routing", Technical Report 2000-010, Dept. of Information Technology, Uppsala University, Uppsala, Sweden, April 2000.
